# Link Discovery using Graph Feature Tracking

**Emile Richard**
ENS Cachan - CMLA & MilleMercis, France
`r.emile.richard@gmail.com`

**Nicolas Baskiotis**
ENS Cachan - CMLA
`nicolas.baskiotis@lip6.com`

**Theodoros Evgeniou**
Technology Management and Decision Sciences,
INSEAD
Bd de Constance, Fontainebleau 77300, France
`theodoros.evgeniou@insead.edu`

**Nicolas Vayatis**
ENS Cachan & UniverSud - CMLA UMR CNRS 8536, France
`nicolas.vayatis@cmla.ens-cachan.fr`

## Abstract

We consider the problem of discovering links of an evolving undirected graph given a series of past snapshots of that graph. The graph is observed through the time sequence of its adjacency matrix and only the presence of edges is observed. The absence of an edge on a certain snapshot cannot be distinguished from a missing entry in the adjacency matrix. Additional information can be provided by examining the dynamics of the graph through a set of topological features, such as the degrees of the vertices. We develop a novel methodology by building on both static matrix completion methods and the estimation of the future state of relevant graph features. Our procedure relies on the formulation of an optimization problem which can be approximately solved by a fast alternating linearized algorithm whose properties are examined. We show experiments with both simulated and real data which reveal the interest of our methodology.

## 1 Introduction

The prediction of the future state of an evolving graph is a challenge of interest in many applications such as predicting hyperlinks of webpages [16], finding protein-protein interactions [7], studying social networks [9], as well as collaborative filtering and recommendations [6]. Link prediction can also be seen as a special case of matrix completion where the goal is to estimate the missing entries of the adjacency matrix of the graph where the entries can be only "0s" and "1s". Matrix completion became popular after the Netflix Challenge and has been extensively studied on both theoretical and algorithmic aspects [15]. In this paper we consider a special case of predicting the evolution of a graph, where we only predict the new edges given a fixed set of vertices of an undirected graph by using the dynamics of the graph over time.

Most of the existing methods in matrix completion assume that weights over the entries (i.e. the edges of the graph, e.g. scores in movie recommendation applications) are observed [3]. These weights provide a richer information than the binary case (existence or absence of a link). Consider for instance the issue of link prediction in recommender systems. In that case, we consider a bipartite graph for which the vertices represent products and users, and the edges connect users with the products they have purchased in the past. The setup we consider in the present paper corresponds to

the binary case where we only observe purchase data, say the presence of a link in the graph, without any score or feedback on the product for a given user. Hence, we will deal here with the situation where the components of snapshots of the adjacency matrix only consist of "1s" and missing values.

Moreover, link prediction methods typically use only one snapshot of the graph's adjacency matrix - the most recent one - to predict its missing entries [9], or rely on latent variables providing semantic information for each vertex [11]. Since these methods do not use any information over time, they can be called *static methods*. Static methods are based on the heuristic that some topological features, such as the degree, the clustering coefficient, or the length of the paths, follow specific distributions. However, information about how the links of the graph and its topological features have been evolving over time may also be useful to predict future links. In the example of recommender systems, knowing that a particular product has been purchased by increasingly more people in a short time window provides useful information about the type of the recommendations to be made in the next period. The main idea underlying our work lies in the observation that a few graph features can capture the dynamics of the graph evolution and provide information for predicting future links.

The purpose of the paper is to present a procedure which exploits the dynamics of the evolution of the graph to find unrevealed links in the graph. The main idea is to learn over time the evolution of well-chosen local features (at the level of the vertices) of the graph and then, use the predicted value of these features on the next time period to discover the missing links. Our approach is related to two theoretical streams of research: matrix completion and diffusion models. In the latter only the dynamics over time of the degree of a particular vertex of the graph are modeled - the diffusion of the product corresponding to that vertex for example [17, 14]. Beyond the large number of static matrix completion methods, only a few methods have been developed that combine static and dynamic information mainly using parametric methods – see [4] for a survey. For example, [13] embeds graph vertices on a latent space and use either a Markov model or a Gaussian one to track the position of the vertices in this space; [10] uses a probabilistic model of the time interval between the appearance of two edges or subgraphs to predict future edges or subgraphs. However, to the best of our knowledge, there has not been any regularization based method for this problem, which we consider in this paper.

The setup of dynamic feature-based matrix completion is presented in Section 2. In Section 3, we develop a fast linearized algorithm for efficient link prediction. We then discuss the use and estimation of relevant features within this regularization approach in Section 4. Eventually, numerical experiments on synthetic and real data sets are depicted in Section 5.

## 2  Dynamic feature-based matrix completion

*Setup.* We consider a sequence of $T$ undirected graphs with $n$ vertices and $n \times n$ binary adjacency matrices $A_t, t \in \{1, 2, ..., T\}$ where for each $t$ the edges of the graph are also contained in the graph at time $t + 1$. Given $A_t, t \in \{1, 2, ..., T\}$ the goal is to predict the edges of the graph that are most likely to appear at time $T + 1$, that is, the most likely non-zero elements of the binary adjacency matrix $A_{T+1}$. To this purpose we want to learn an $n \times n$ real-valued matrix $S$ whose elements indicate how likely it is that there is a non-zero value at the corresponding position of matrix $A_{T+1}$. The edges that we predict to be the most likely ones at time $T + 1$ are the ones corresponding to the largest values in $S$.

We assume that certain features of matrices $A_t$ evolve over time smoothly. Such an assumption is necessary to allow learnability of the evolution of $A_t$ over time. For simplicity we consider a linear feature map $f : A_t \mapsto F_t$ where $F_t$ is an $n \times k$ matrix of the form $F_t = A_t \Phi$, with $\Phi$ an $n \times k$ matrix of features. Various feature maps, possibly nonlinear, can be used. We discuss an example of such features $\Phi$ and a way to predict $F_{T+1}$ given past values of the feature map $F_1, F_2, ..., F_T$ in Section 4 – but other features or prediction methods can be used in combination with the main part of the proposed approach. In the proposed method discussed in Section 3 we assume for now that we already have an estimate of $F_{T+1}$.

*An optimization problem.* The procedure we propose for link prediction is based on the assumption that the dynamics of graph features also drive the discovery of the location of new links. Given the last adjacency matrix $A_T$, a set of features $\Phi$, and an estimate $\widehat{F}$ of $F_{T+1}$ based on the sequence

of adjacency matrices $A_t, t \in \{1, 2, ..., T\}$, we want to find a matrix $S$ which fulfills the following requirements:

- $S$ has low rank - this is a standard assumption in matrix completion problems [15].
- $S$ is close to the last adjacency matrix $A_T$ - the distance between these two matrices will provide a proxy for the training error.
- The values of the feature map at $S$ and $A_{T+1}$ are similar.

For any matrix $M$, we denote by $\|M\|_\mathrm{F} = \sqrt{\mathrm{Tr}(M'M)}$ , the Frobenius norm of $M$, with $M'$ being the transpose of $M$ and the trace operator $\mathrm{Tr}(N)$ computes the sum of the diagonal elements of the square matrix $N$. We also define $\|M\|_* = \sum_{k=1}^{n} \sigma_k(M)$ , the nuclear norm of a square matrix $M$ of size $n \times n$, where $\sigma_k(M)$ denotes the $k$-th largest singular value of $M$. We recall that a singular value of matrix $M$ corresponds to the square root of an eigenvalue of $M'M$ ordered decreasingly.

The proposed optimization problem for feature-based matrix completion is then:

$$\min_S L(S, \tau, \nu) , \quad \text{with} \quad L(S, \tau, \nu) \doteq \tau\|S\|_* + \frac{1}{2}\|S - A_T\|_\mathrm{F}^2 + \frac{1}{2}\nu\|S\Phi - \widehat{F}\|_\mathrm{F}^2 , \quad (1)$$

and where $\tau$ and $\nu$ are positive regularization parameters. Each term of the functional $L$ reflects the aforementioned requirements for the desired matrix $S$. In the case where $\nu = 0$, we do not use information about the dynamics of the graph. The minimizer of $L$ corresponds to the singular value thresholding approach developed in [2], which is therefore a special case of (1). Note that a key difference between link prediction and matrix completion is that in (1) the training error uses all entries of the adjacency matrix while in the case of matrix completion only the known entries (in our case the "1s") are used. We now discuss an efficient optimization algorithm for (1), the main part of this work.

## 3  An algorithm for link discovery

Solving (1) is computationally slow. We adapt the fast linearization method developed in [5] to our problem, which attains an optimal iteration complexity when using only first order information. Here, the functional $L(S, \tau, \nu)$ is continuous and convex but not differentiable with respect to $S$. We propose to convert the minimization of the target functional $L(S, \tau, \nu)$ into a tractable problem through the following steps:

1. *Variable splitting* - Set:

$$g(S, \tau) = \tau\|S\|_* \qquad \text{and} \qquad h(S, \nu) = \frac{1}{2}\|S - A_T\|_\mathrm{F}^2 + \frac{1}{2}\nu\|S\Phi - \widehat{F}\|_\mathrm{F}^2 .$$

   Denote by $S, \widetilde{S}$ two $n \times n$ matrices. Then, the optimization problem (1) is equivalent to:

$$\min_{S, \widetilde{S}} \mathcal{L}(S, \widetilde{S}) , \qquad \text{subject to } S - \widetilde{S} = 0 . \quad (2)$$

   where $\mathcal{L}(S, \widetilde{S}) \doteq g(S, \tau) + h(\widetilde{S}, \nu)$.

2. *Smoothing the nuclear norm* - We recall the variational formulation of the nuclear norm $\|S\|_* = \max_Z\{\langle S, Z \rangle \ : \ \sigma_1(Z) \leq 1\}$. Using the technique from [12], we can use a smooth approximation of the nuclear norm and replace $g$ in the functional by a surrogate function $g_\eta$ with $\eta > 0$ being a smoothing parameter:

$$g_\eta(S, \tau) = \tau \cdot \max_Z \left\{ \langle S, Z \rangle - \frac{\eta}{2}\|Z\|_\mathrm{F}^2 \ : \ \sigma_1(Z) \leq 1 \right\}$$

3. *Alternating minimization* - We propose to minimize the functional which is continuous, differentiable and convex:

$$\mathcal{L}_\eta(S, \widetilde{S}) \doteq g_\eta(S, \tau) + h(\widetilde{S}, \nu) , \quad (3)$$

   under the constraint that $S = \widetilde{S}$. To do this, one has to minimize simultaneously the two functions $g_\eta$ and $h$. In order to derive the iterative algorithm based on linearized alternating

minimization, we introduce two strictly convex approximations of these functions which involve an additional parameter $\mu > 0$:

$$G_{\eta,\mu}(S, \tilde{S}) = g_\eta(S, \tau) + \langle \nabla h(\widetilde{S}), S - \widetilde{S} \rangle + \frac{1}{2\mu} \|S - \widetilde{S}\|_{\mathrm{F}}^2$$

$$H_\mu(S, \widetilde{S}) = h(\widetilde{S}, \nu) + \langle \nabla g_\eta(S), \widetilde{S} - S \rangle + \frac{1}{2\mu} \|S - \widetilde{S}\|_{\mathrm{F}}^2$$

where $\langle B, C \rangle = \mathrm{Tr}(B'C)$ for two matrices $B$, $C$. The tuning of the parameter $\mu$ will be discussed with the convergence results at the end of this section. We denote by $m^G(\widetilde{S})$ the minimizer of $G_{\eta,\mu}(S, \tilde{S})$ with respect to $S$ and $m^H(S)$ the minimizer of $H_\mu(S, \widetilde{S})$ with respect to $\widetilde{S}$. We can now formulate an algorithm for the fast minimization of the functional $\mathcal{L}_\eta(S, \widetilde{S})$ inspired by the algorithm FALM in [5] (see **Algorithm 1**). Note that, in the alternating descent for the simultaneous minimization of the two functions $G_{\eta,\mu}$ and $H_\mu$, we use an auxiliary matrix $Z_k$. This matrix is a linear combination of the updates for $S$ and $\widetilde{S}$. The work by Ma and Goldfarb shows indeed that the particular choice made here leads to an optimal rate of numerical convergence. Key formulas in the link prediction algorithm are those of the minimizers $m^G(\widetilde{S})$ and $m^H(S)$. It turns out that in our case, these minimizers have explicit expressions which can be derived when solving the first-order optimality condition as **Proposition 1** shows.

---

**Algorithm 1** - Link Discovery Algorithm

> **Parameters**: $\tau, \nu, \eta$
>
> **Initialization**: $W_0 = Z_1 = A_T$, $\alpha_1 = 0$
>
> **for** $k = 1, 2, \dots$ **do**
>
> $\quad S_k \leftarrow m^G(Z_k) \quad$ and $\quad \widetilde{S}_k \leftarrow m^H(S_k)$
>
> $\quad W_k \leftarrow \frac{1}{2}(S_k + \widetilde{S}_k)$
>
> $\quad \alpha_{k+1} \leftarrow \frac{1}{2}(1 + \sqrt{1 + 4\alpha_k^2})$
>
> $\quad Z_{k+1} \leftarrow W_k + \frac{1}{\alpha_{k+1}}\left( \alpha_k(\widetilde{S}_k - W_{k-1}) - (W_k - W_{k-1}) \right)$
>
> **end for**

---

**Proposition 1** *Let $\hat{S} = \widetilde{S} - \mu \nabla h(\widetilde{S})$ and the singular value decomposition $\widehat{S} = \widehat{U} Diag(\widehat{\sigma})\widehat{V}$. We also consider the singular value decomposition of $S$ denoted by $S = U Diag(\eta\lambda)V$. We set the notation, for $x > 0$:*

$$\alpha(x) = \max \left\{ \frac{x}{1 + \dfrac{\tau\mu}{\eta}}, x - \tau\mu \right\}$$

*We then have:*

$$m^G(\widetilde{S}) = \widehat{U} Diag\{\alpha(\hat{\sigma})\}\widehat{V}$$

$$m^H(S) = \left( A_T - \tau U Diag(\min\{\lambda, 1\})V + \frac{1}{\mu}S + \nu \widehat{F}\Phi' \right) \left( \left(1 + \frac{1}{\mu}\right) I_n + \nu \Phi\Phi' \right)^{-1}.$$

The proof can be found in the Appendix.

*Validity of the approximations and rates of convergence.* Our strategy replaces the non-differentiable term in $\mathcal{L}$ by a smooth version of it. The next result offers guarantees that minimizing the surrogate function (3) provides an approximate solution of the initial problem (1). We will say that an element $x_\epsilon$ is an $\epsilon$-optimal solution of a function $\Psi(x)$ if it is such that $\Psi(x_\epsilon) \leq \inf_x \Psi(x) + \epsilon$.

**Proposition 2** *The following statements hold true:*

- *We have, for any $(S, \widetilde{S})$:*

$$\mathcal{L}_\eta(S, \widetilde{S}) \leq \mathcal{L}(S, \widetilde{S}) \leq \mathcal{L}_\eta(S, \widetilde{S}) + \frac{n\eta}{2} \ .$$

- *To find an $\epsilon$-optimal solution of $\mathcal{L}(S, \widetilde{S})$, it suffices to find an $\epsilon/2$-optimal solution of $\mathcal{L}_\eta(S, \widetilde{S})$ with $\eta = \epsilon/n$.*

The proof of this result can be derived straightforwardly from [5]. Moreover, following the proof of Theorem 4.3 in [5], one can show that the number of iterations in order to reach an $\epsilon$-optimal solution of $\mathcal{L}_\eta$ using **Algorithm 1** is of the order $O(1/\sqrt{\epsilon})$. In that result of [5], an optimal choice of the parameter $\mu$ is provided as the inverse of the largest value for the Lipschitz constant of each of the gradients of $g_\eta$ and $h$. With our notations, we can easily derive here: $\mu = \min\big(\eta/\tau, 1/(1 + \nu\sigma_1(\Phi))\big)$, where $\sigma_1(\Phi)$ is the largest singular value of $\Phi$.

## 4   Learning the graph features

As discussed above one can use various features $\Phi$ and methods to predict the $n \times k$ matrix $F_{T+1}$ given past values of the feature map $F_1, F_2, ..., F_T$. We consider a particular case here to use in conjunction with the main algorithm in the previous section. In particular, we use as features $\Phi$ the first $k$ eigenvectors of the adjacency matrix $A_T$. Let $A_T = \Omega\Lambda\Omega'$ be the orthonormal eigenvalue decomposition of $A_T$ which is symmetric. We set $\Phi = \Omega_{(:,1:k)}\Lambda_{(:,1:k)}^{-1}$, an $n \times k$ matrix. Note that $A_T\Phi = \Omega_{(:,1:k)}$ and that $\Omega_{(:,1:k)}$ is the most informative $n \times k$ matrix for the reconstruction of $A_T$. The suggested method aims to estimate $A_{T+1}\Phi$ that is informative for the reconstruction of $A_{T+1}$. We denote by $\Phi_j$, $j \in \{1, 2, ..., k\}$ the $n$-dimensional feature vectors which are the columns of $\Phi$.

For each feature $j \in \{1, 2, ..., k\}$, we consider the $n$-dimensional time series $\{A_t\Phi_j \, , \, t = 1, \ldots, T\}$ which describes the evolution of the $j$-th feature over the $n$ vertices of the graph. We now describe the procedure for learning the evolution of this $j$-th graph feature over time:

1. Fix an integer $m < T$ to learn a map between $m$ past values $(A_{t-m}\Phi_j, \ldots, A_{t-1}\Phi_j)$ and the current value of the $n$-dimensional vector $A_t\Phi_j$.

2. Construct the training data for the learning step by using a sliding window of size $m$ from time $t = 1$ to $t = T$; we then have $T - m + 1$ training data of dimension $n \times m$ for each feature $j$.

3. Use ridge regression to fit the training data.

4. Estimate the $j$-th column of $F_{T+1}$ as the predicted value for $A_{T+1}\Phi_j$ using the regression model at the "point" $(A_{T-m+1}\Phi_j, \ldots, A_T\Phi_j)$.

Collecting the results for each $j \in \{1, 2, ..., k\}$, we obtain the estimate $\widehat{F}$ for the matrix $F_{T+1}$ used in (1). We point out that using construction with a time shift means that implicitly the relation between $m$ consecutive values of $(A_{t-m}\Phi_j, \ldots, A_{t-1}\Phi_j)$ and the next value $A_t\Phi_j$ is stable over time (stationarity assumption). Clearly methods other than ridge regression or other ways of creating the training data can be used, which we leave for future work.

## 5   Experimental Results

We tested the proposed method using both simulated and real data sets. As benchmarks we use the following methods:

1. Static matrix completion corresponding to $\nu = 0$ in (1).

2. The Katz algorithm [8] considered as one of the best static link prediction methods.

3. The Preferential Attachment method [1] for which the score ("likelihood") of an edge $\{u, v\}$ is $d_u d_v$ where $d_u$ and $d_v$ are the degrees of $u$ and $v$.

## 5.1 Synthetic Data

We generate sequences of graphs as follows. We first generate a sequence of $T$ matrices $Q(t)$ of size $n \times r$ whose entries $Q_{i,j}(t)$ are increasing over time as a sigmoid function :

$$Q_{i,j}(t) = \frac{1}{2} \left( 1 + \text{erf} \left( \frac{t - \mu_{i,j}}{\sqrt{2\sigma_{i,j}^2}} \right) \right)$$

where $\mu_{i,j} \in [0; T]$, $\sigma_{i,j} \in [0; T/3]$ are picked uniformly for each $(i, j)$. These matrices provide a synthetic model for the evolution of the graph over time. We then add noise to the time dynamics as follows. For a given noise level $\omega \in [0, 1]$ we replace each entry of $Q_{i,j}(t)$ with probability $\omega$ with any of the other values $Q_{i,j}(s)$ for $s$ picked uniformly from $\{1, 2, ..., T\}$. Having constructed the matrices $Q(t)$, we then generate matrices $S(t) = Q(t)Q(t)'$ which are of rank $r$. We finally generate the adjacency matrix $A_t$ as

$$A(t) = \mathbf{1}_{[\theta;\infty[}(S(t))$$

for a threshold $\theta$. We pick $\theta$ so that the sparsity (i.e. proportion of non-zero entries) of $A_T$ reflects the sparsity of the real data used in the next section ($\approx 10^{-3}$). In the experiments, we simulated graphs with $n = 1000$ vertices.

## 5.2 Real Data

**Collaborative Filtering**[1]   We can see the purchase histories of e-commerce websites as graph sequences where links are established between a user and a product when the user purchases that product. We use data from 10 months music purchase history of a major e-commerce website to evaluate our method. For our test we selected a set of $10^3$ users and $10^3$ products that had the highest degrees (number of sales). We split the $8.5 \times 10^3$ edges of the graph (corresponding to purchases) into two parts following their occurrence time. We used the data of the 8 first months to predict the features at the end of the 10th month and use these features as well as the matrix at the end of the 8th month to discover the purchases during the 2 last months.

## 5.3 Results

The results are shown in Figure 1 and Tables 1 and 2. The Area Under an ROC Curve (AUC) is reported. For the simulation data we report the average AUC over 10 simulation runs.

From the simulation results we observe that for low rank underlying matrices, our method outperforms the rivals. The same comparative results were observed for ranks as high as 100. Our method (as well as the static low rank method based on the low rank hypothesis) however fails when the rank of $S(t)$ is high. However, even in this case our method outperforms the method of static matrix completion.

The results with the real data further indicate the advantage of using information about the evolution of the graph over time. Similarly to the simulation data, the proposed method outperforms the static matrix completion one.

# 6 Conclusion

The main contribution of this work is the formulation of a learning problem that can be used to predict the evolution of the edges of a graph over time. A regularization approach to combine both static graph information as well as information about the dynamics of the evolution of the graph over time is proposed and an optimization algorithm is developed. Despite using simple graph features

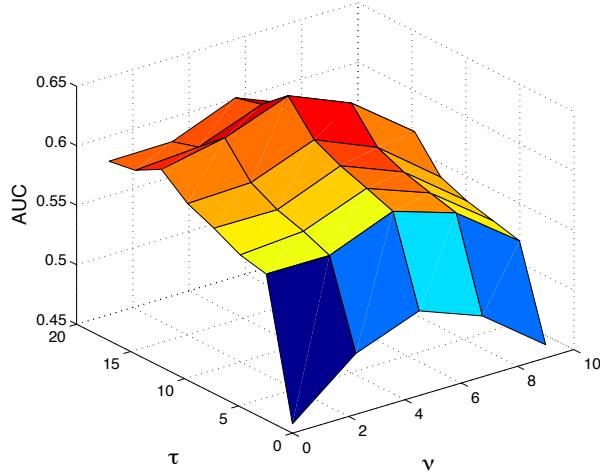

Figure 1: AUC performance of the proposed algorithm with respect to the two parameters $\tau$ and $\nu$ on simulated data.

| $(r,\omega) \setminus$ Method | Proposed Method | Static | Pref. A. | Katz |
|---|---|---|---|---|
| (5,0.000) | **0.671**±0.008 | $0.648 \pm 0.008$ | $0.627 \pm 0.015$ | $0.616 \pm 0.015$ |
| (5,0.250) | **0.675** $\pm$ 0.009 | $0.642 \pm 0.007$ | $0.602 \pm 0.016$ | $0.592 \pm 0.016$ |
| (5,0.750) | $0.519 \pm 0.007$ | **0.525** $\pm$ 0.005 | $0.497 \pm 0.007$ | $0.491 \pm 0.007$ |
| (500,0.000) | $0.592 \pm 0.008$ | $0.587 \pm 0.007$ | **0.671** $\pm$ 0.010 | $0.667 \pm 0.009$ |
| (500,0.250) | $0.607 \pm 0.011$ | $0.588 \pm 0.009$ | **0.649** $\pm$ 0.009 | $0.643 \pm 0.009$ |
| (500,0.750) | $0.601 \pm 0.010$ | $0.583 \pm 0.007$ | **0.645** $\pm$ 0.017 | $0.641 \pm 0.017$ |

Table 1: Simulation data. The average AUC over 10 simulation runs is reported. For each row the pair of numbers in the first column show the rank $r$ and the noise level $\omega$.

as well as estimation of the evolution of the feature values over time, experiments indicate that the proposed optimization method improves performance relative to benchmarks. Testing, or learning, other graph features as well as other ways to model their dynamics over time may further improve performance and is part of future work.

## Appendix - Proof of Proposition 1

We first write the optimality condition for $G_{\eta,\mu}(S, \tilde{S})$ with respect to $S$:

$$\nabla_S g_\eta(S) + \nabla h(\tilde{S}) + \frac{1}{\mu}(S - \tilde{S}) = 0 \ .$$

| $\tau \setminus \nu$ | 0 | 0.1 | 0.3 | 0.7 | 1.6 |
|---|---|---|---|---|---|
| 0 | 0.568 | 0.584 | 0.585 | 0.585 | 0.562 |
| 1 | 0.626 | 0.684 | 0.683 | 0.675 | 0.668 |
| 2 | 0.638 | 0.678 | 0.671 | 0.688 | 0.672 |
| 3 | 0.569 | 0.646 | 0.635 | 0.645 | 0.643 |
| 4 | 0.569 | 0.556 | 0.562 | 0.565 | 0.563 |

Table 2: Collaborative Filtering data; AUC for different values of $\tau$ and $\nu$. The AUC of preferential attachment is 0.6019, and Katz reaches 0.6670

With the notations for $\widehat{S}$, the previous condition can be written:

$$\mu \nabla g_\eta(S) + S - \widehat{S} = 0 \,.$$

We now use the fact that $\nabla g_\eta(S) = \tau U \mathrm{Diag}(\min\{\gamma, 1\})V$ where $S/\eta = U \mathrm{Diag}(\gamma)V$ (see [5]). This observation leads to the solution where $S$ satisfies:

$$U = \widehat{U} \,, \quad V = \widehat{V} \,, \quad \text{and} \quad \hat{\sigma} = \mu\tau \min\{\gamma, 1\} + \eta\gamma \,,$$

which gives the first result, since there is a unique solution due to the strict convexity of the function.

Similarly, the optimality condition of $H_\mu(S, \widetilde{S})$ with respect to $\widetilde{S}$ is

$$\nabla h(\widetilde{S}) + \nabla g_\eta(S) + \frac{1}{\mu}(\tilde{S} - S) = 0 \,.$$

Since the function $h$ is differentiable as the sum of two quadratic terms, we have:

$$\nabla h(\widetilde{S}) = \widetilde{S} - A_T + \nu(\widetilde{S}\Phi - \widehat{F})\Phi' \,,$$

and we can derive the optimal value for $\widetilde{S}$:

$$m^H(S) = \left( A_T - \nabla g_\eta(S) + \frac{1}{\mu}S + \nu\widehat{F}\Phi' \right) \left( \left(1 + \frac{1}{\mu}\right) I_n + \nu\Phi\Phi' \right)^{-1} \,. \quad \square$$

**Acknowledgments**

This work was partially supported by Digitéo (Bémol project), that authors greatly thank.

## Footnotes

[1]Notice that we are looking to discover only unobserved links and not new occurences of past links. Thus the comparaison with some popular benchmarks (as coauthorship data sets) is inappropriate.

## References

[1] A. L. Barabási, H. Jeong, Z. Nda, A. Schubert, and T. Vicsek. Evolution of the social network of scientific collaborations. *Physica A: Statistical Mechanics and its Applications*, 311(3-4):590–614, 2002.

[2] Emmanuel J. Candès and Terence. Tao. A singular value thresholding algorithm for matrix completion. *SIAM Journal on Optimization*, 20(4):1956–1982, 2008.

[3] Emmanuel J. Candès and Terence Tao. The power of convex relaxation: Near-optimal matrix completion. *IEEE Transactions on Information Theory*, 56(5), 2009.

[4] Lise Getoor and Christopher P. Diehl. Link mining: a survey. *SIGKDD Explorations Newsletter*, 7(2):3–12, 2005.

[5] Donald Goldfarb and Shiqlan Ma. Fast alternating linearization methods for minimizing the sum of two convex functions. *Technical Report, Department of IEOR, Columbia University*, 2009.

[6] Yifan Hu, Yehuda Koren, and Chris Volinsky. Collaborative filtering for implicit feedback datasets. In *Proceedings of the 8th IEEE International Conference on Data Mining (ICDM 2008)*, pages 263–272, 2008.

[7] Hisashi Kashima, Tsuyoshi Kato, Yoshihiro Yamanishi, Masashi Sugiyama, and Koji Tsuda. Link propagation: A fast semi-supervised learning algorithm for link prediction. In *Proceedings of the SIAM International Conference on Data Mining, SDM 2009*, pages 1099–1110, 2009.

[8] Leo Katz. A new status index derived from sociometric analysis. *Psychometrika*, 18(1):39–43, 1953.

[9] David Liben-Nowell and Jon Kleinberg. The link-prediction problem for social networks. *Journal of the American Society for Information Science and Technology*, 58(7):1019–1031, 2007.

[10] Tanya Y. Berger-Wolf Mayank Lahiri. Structure prediction in temporal networks using frequent subgraphs. *IEEE Symposium on Computational Intelligence and Data Mining (CIDM)*, 2007.

[11] Kurt Miller, Thomas Griffiths, and Michael Jordan. Nonparametric latent feature models for link prediction. In Y. Bengio, D. Schuurmans, J. Lafferty, C. K. I. Williams, and A. Culotta, editors, *Advances in Neural Information Processing Systems 22*, pages 1276–1284. 2009.

[12] Yu Nesterov. Smooth minimization of non-smooth functions. *Mathematical Programming*, 103(1):127–152, 2005.

[13] Purnamrita Sarkar, Sajid Siddiqi, and Geoffrey J. Gordon. A latent space approach to dynamic embedding of cooccurrence data. In *In Proceedings of the Eleventh International Conference on Artificial Intelligence and Statistics (AI-STATS)*, 2007.

[14] Ashish Sood, Gareth M. James, and Gerard J. Tellis. Functional regression: A new model for predicting market penetration of new products. *Marketing Science*, 28(1):36–51, 2009.

[15] Nathan Srebro, Jason D. M. Rennie, and Tommi S. Jaakkola. Maximum-margin matrix factorization. In Lawrence K. Saul, Yair Weiss, and Léon Bottou, editors, *Advances in Neural Information Processing Systems 17*, pages 1329–1336. MIT Press, Cambridge, MA, 2005.

[16] Ben Taskar, Ming-Fai Wong, Pieter Abbeel, and Daphne Koller. Link prediction in relational data. In Sebastian Thrun, Lawrence Saul, and Bernhard Schölkopf, editors, *Advances in Neural Information Processing Systems 16*. MIT Press, Cambridge, MA, 2004.

[17] Demetrios Vakratsas, Fred M. Feinberg, Frank M. Bass, and Gurumurthy Kalyanaram. The Shape of Advertising Response Functions Revisited: A Model of Dynamic Probabilistic Thresholds. *Marketing Science*, 23(1):109–119, 2004.

